# A Rigorous Analysis Of Linsker-type Hebbian Learning

**J. Feng**
Mathematical Department
University of Rome "La Sapienza"
P. le A. Moro, 00185 Rome, Italy
feng@mat.uniroma1.it

**H. Pan    V. P. Roychowdhury**
School of Electrical Engineering
Purdue University
West Lafayette, IN 47907
hpan@ecn.purdue.edu
vwani@drum.ecn.purdue.edu

## Abstract

We propose a novel rigorous approach for the analysis of Linsker's unsupervised Hebbian learning network. The behavior of this model is determined by the underlying nonlinear dynamics which are parameterized by a set of parameters originating from the Hebbian rule and the arbor density of the synapses. These parameters determine the presence or absence of a specific receptive field (also referred to as a 'connection pattern') as a saturated fixed point attractor of the model. In this paper, we perform a qualitative analysis of the underlying nonlinear dynamics over the parameter space, determine the effects of the system parameters on the emergence of various receptive fields, and predict precisely within which parameter regime the network will have the potential to develop a specially designated connection pattern. In particular, this approach exposes, for the first time, the crucial role played by the synaptic density functions, and provides a complete precise picture of the parameter space that defines the relationships among the different receptive fields. Our theoretical predictions are confirmed by numerical simulations.

# 1   Introduction

For the purpose of understanding the self-organization mechanism of primary vi-
sual system, Linsker has proposed a multilayered unsupervised Hebbian learning
network with random uncorrelated inputs and localized arborization of synapses
between adjacent layers (Linsker, 1986 & 1988). His simulations have shown that
for appropriate parameter regimes, several structured connection patterns (e.g.,
centre-surround and oriented afferent receptive fields (aRFs)) occur progressively
as the Hebbian evolution of the weights is carried out layer by layer. The behavior
of Linsker's model is determined by the underlying nonlinear dynamics which are
parameterized by a set of parameters originating from the Hebbian rule and the
arbor density of the synapses. For a nonlinear system, usually, there *coexist* several
attractors for the same set of system parameters. That is, for a given set of the
parameters, the state space comprises several attractive basins, each corresponding
to a steady state respectively. The initial condition determines which attractor will
be eventually reached. At the same time, a nonlinear system could have a different
group of coexisting attractors for a different set of system parameters. That is,
one could make the presence or absence of a specific state as a fixed point attrac-
tor by varying the set of the parameters. For a development model like Linsker's
network, what is expected to be observed is that the different aRFs could emerge
under different sets of parameters but should be relatively not sensitive to the initial
conditions. In other words, the dynamics should avoid the coexistence of several
attractors in an appropriate way. The purpose of this paper is to gain more insights
into the dynamical mechanism of this self-organization model by performing a rig-
orous analysis on its parameter space without any approximation. That is, our goal
is to reveal the effects of the system parameters on the stability of aRFs, and to
predict *precisely* within which parameter regime the network will have the poten-
tial to develop a specially designated aRF. The novel rigorous approach presented
here applies not only to the Linsker-type Hebbian learning but also to other related
self-organization models about neural development.

In Linsker's network, each cell in the present layer $\mathcal{M}$ receives synaptic inputs from
a number of cells in the preceding layer $\mathcal{L}$. The density of these synaptic connections
decreases monotonically with distance $r_{\mathcal{L}}$ from the point underlying the $\mathcal{M}$-cell's
position. Since the synaptic weights change on a long time scale compared to the
variation of random inputs, by averaging the Hebb rule over the ensemble of inputs
in layer $\mathcal{L}$, the dynamical equation for the development of the synaptic strength
$\omega_\tau(i)$ between a $\mathcal{M}$-cell and $i$-th $\mathcal{L}$-cell at time $\tau$ is

$$\omega_{\tau+1}(i) = f\{\omega_\tau(i) + k_1 + \sum_{j=1}^{N_{\mathcal{L}}}[Q_{ij}^{\mathcal{L}} + k_2]r(j)\omega_\tau(j)\} \qquad (1)$$

where $k_1, k_2$ are system parameters which are particular combinations of the con-
stants of the Hebb rule, $r(\cdot)$ is a non-negative normalized synaptic density function
(SDF) [1], and $\sum_{i \in \mathcal{L}} r(i) = 1$, and $f(\cdot)$ is a limiter function defined by $f(x) = \omega_{max}$,
if $x > \omega_{max}$; $= x$, if $|x| \leq \omega_{max}$; and $= -\omega_{max}$, if $x < -\omega_{max}$. The covariance

Linsker's formulation. A rigorous explanation for this equivalence is given in MacKay
& Miller, 1990.

matrix $\{Q_{ij}\}$ of the layer $\mathcal{L}$ describes the correlation of activities of the $i$-th and the $j$-th $\mathcal{L}$-cells. Actually, the covariance matrix of each layer is determined by SDFs $r(\cdot)$ of all layers preceding the layer under consideration.

The idea of this paper is the following. It is well known that in general it is intractable to characterize the behavior of a nonlinear dynamics, since the nonlinearity is the cause of the coexistence of many attractors. And one has the difficulty in obtaining the complete characteristics of attractive basins in the state space. But usually for some cases, it is relatively easy to derive a necessary and sufficient condition to check whether a given state is a fixed point of the dynamics. In terms of this condition, the *whole* parameter regime for the emergence of a fixed point of the dynamics may be obtained in the parameter space. If we are further able to prove the stability of the fixed point, which implies that this fixed point is a steady state if the initial condition is in a nonempty vicinity in the state space, we can assert the occurrence of this fixed point attractor in that parameter regime. For Linsker's network, fortunately, the above idea can be carried out because of the specific form of the nonlinear function $f(\cdot)$. Due to space limitations, the rigorous proofs are in (Feng, Pan, & Roychowdhury, 1995).

## 2   The Set Of Saturated Fixed Point Attractors And The Criterion For The Division Of Parameter Regimes

In fact, Linsker's model is a system of first-order nonlinear difference equations, taking the form

$$\omega_{\tau+1}(i) = f[\omega_\tau(i) + h_i(\omega_\tau, k_1, k_2)], \qquad \omega_\tau = \{\omega_\tau(j), j = 1, ..., N_{\mathcal{L}}\}, \qquad (2)$$

where $h_i(\omega_\tau, k_1, k_2) = k_1 + \sum_{j=1}^{N_{\mathcal{L}}}[Q_{ij}^{\mathcal{L}} + k_2]r(j)\omega_\tau(j)$. And the aRFs observed in Linsker's simulation are the *saturated fixed point attractors* of this nonlinear system (2). Since the limiter function $f(\cdot)$ is defined on a hypercube $\Omega = [-\omega_{max}, \omega_{max}]^{N_{\mathcal{L}}}$ in weight state space within which the dynamics is dominated by the linear system $\omega_{\tau+1}(i) = \omega_\tau(i) + h_i(\omega_\tau, k_1, k_2)$, the short-time behaviors of evolution dynamics of connection patterns can be fully characterized in terms of the properties of eigenvectors and their eigenvalues. But this method of stability analysis will not be suitable for the long-time evolution of equation (1) or (2), provided the hypercube constraint is reached as the first largest component of $\omega$ reaches saturation. However, it is well-known that a *fixed point* or an *equilibrium state* of dynamics (2) satisfies

$$\omega_\tau(i) = f[\omega_\tau(i) + h_i(\omega_\tau, k_1, k_2)]. \qquad (3)$$

Because of the special form of the nonlinear function $f(\cdot)$, the fixed point equation (3) implies that $\exists \mathcal{T}$, such that for $\tau > \mathcal{T}$,

$$|\omega_\tau(i) + h_i(\omega_\tau, k_1, k_2)| \geq \omega_{max},$$

if $h_i(\omega, k_1, k_2) \neq 0$. So a saturated fixed point $\omega_\tau(i)$ must have the same sign as $h_i(\omega_\tau, k_1, k_2)$, i.e.

$$\omega_\tau(i)h_i(\omega_\tau, k_1, k_2) > 0.$$

By using the above idea, our Theorems 1 & 2 (proven in Feng, Pan, & Roychowdhury, 1995) state that the set of saturated fixed point attractors of the dynamics in

equation (1) is given by

$$\Omega_{FP} = \{\omega \mid \omega(i)h_i(\omega_\tau, k_1, k_2) > 0,\ 1 \le i \le N_{\mathcal{L}}\},$$

and $\omega \in \Omega_{FP}$ is stable, where the weight vector $\omega$ belongs to the set of all extreme points of the hypercube $\Omega$ (we assume $\omega_{max} = 1$ without loss of generality).

We next derive an explicit necessary and sufficient condition for the emergence of structured aRFs, i.e., we derive conditions to determine whether a given $\omega$ belongs to $\Omega_{FP}$. Define $J^+(\omega) = \{i \mid \omega(i) = 1\}$ as the index set of cells at the preceding layer $\mathcal{L}$ with excitatory weight for a connection pattern $\omega$, and $J^-(\omega) = \{i \mid \omega(i) = -1\}$ as the index set of $\mathcal{L}$-cells with inhibitory weight for $\omega$. Note from the property of fixed point attractors that a connection pattern $\omega$ is an attractor of the dynamics (1) if and only if for $i \in J^+(\omega)$, we have

$$\omega(i)\{k_1 + \sum_j [Q_{ij}^{\mathcal{L}} + k_2]r(j)\omega(j)\} =$$

$$\omega(i)\{k_1 + \sum_{j \in J^+(\omega)}[Q_{ij}^{\mathcal{L}} + k_2]r(j)\omega(j) + \sum_{j \in J^-(\omega)}[Q_{ij}^{\mathcal{L}} + k_2]r(j)\omega(j)\} > 0.$$

By the definition of $J^+(\omega)$ and $J^-(\omega)$, we deduce from the above inequality that

$$k_1 + \sum_{j \in J^+(\omega)} [Q_{ij}^{\mathcal{L}} + k_2]r(j) - \sum_{j \in J^-(\omega)} [Q_{ij}^{\mathcal{L}} + k_2]r(j) > 0$$

namely

$$k_1 + k_2[\sum_{j \in J^+(\omega)} r(j) - \sum_{j \in J^-(\omega)} r(j)] > \sum_{j \in J^-(\omega)} Q_{ij}^{\mathcal{L}}r(j) - \sum_{j \in J^+(\omega)} Q_{ij}^{\mathcal{L}}r(j).$$

Inequality above is satisfied for all $i$ in $J^+(\omega)$, and the left hand is independent of $i$. Hence,

$$k_1 + k_2[\sum_{j \in J^+(\omega)} r(j) - \sum_{j \in J^-(\omega)} r(j)] > \max_{i \in J^+(\omega)}[\sum_{j \in J^-(\omega)} Q_{ij}^{\mathcal{L}}r(j) - \sum_{j \in J^+(\omega)} Q_{ij}^{\mathcal{L}}r(j)].$$

On the other hand, for $i \in J^-(\omega)$, we can similarly deduce that

$$k_1 + k_2[\sum_{j \in J^+(\omega)} r(j) - \sum_{j \in J^-(\omega)} r(j)] < \min_{i \in J^-(\omega)}[\sum_{j \in J^-(\omega)} Q_{ij}^{\mathcal{L}}r(j) - \sum_{j \in J^+(\omega)} Q_{ij}^{\mathcal{L}}r(j)].$$

We introduce the *slope function*:

$$c(\omega) \stackrel{\text{def}}{=} \sum_{j \in J^+(\omega)} r(j) - \sum_{j \in J^-(\omega)} r(j)$$

which is the difference of sums of the SDF $r(\cdot)$ over $J^+(\omega)$ and $J^-(\omega)$, and two $k_1$-*intercept functions*:

$$d_1(\omega) \stackrel{\text{def}}{=} \begin{cases} \max_{i \in J^+(\omega)}(\sum_{j \in J^-(\omega)} Q_{ij}^{\mathcal{L}}r(j) - \sum_{j \in J^+(\omega)} Q_{ij}^{\mathcal{L}}r(j)), & \text{if } J^+(\omega) \ne \emptyset \\ -\infty, & \text{if } J^+(\omega) = \emptyset \end{cases}$$

and

$$d_2(\omega) \stackrel{\text{def}}{=} \begin{cases} \min_{i \in J^-(\omega)}(\sum_{j \in J^-(\omega)} Q_{ij}^{\mathcal{L}}r(j) - \sum_{j \in J^+(\omega)} Q_{ij}^{\mathcal{L}}r(j)), & \text{if } J^-(\omega) \ne \emptyset \\ \infty, & \text{if } J^-(\omega) = \emptyset \end{cases}$$

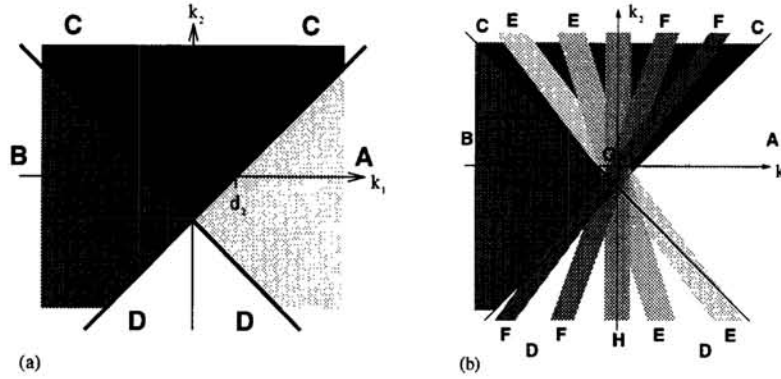

Figure 1: The parameter subspace of $(k_1, k_2)$. (a) Parameter regime of $(k_1, k_2)$ to ensure the emergence of all-excitatory (regime A) and all-inhibitory (regime B) connection patterns. The dark grey regime C is the coexistence regime for both all-excitatory and all-inhibitory connection patterns. And the regime D without texture are the regime that Linsker's simulation results are based on, in which both all-excitatory and all-inhibitory connection patterns are no longer an attractor. (b) The principal parameter regimes.

Now from our Theorem 3 in Feng, Pan, & Roychowdhury, 1995, for every layer of Linsker's network, the new rigorous criterion for the division of stable parameter regimes to ensure the development of various structured connection patterns is

$$d_2(\omega) > k_1 + c(\omega)k_2 > d_1(\omega).$$

That is, for a given SDF $r(\cdot)$, the parameter regime of $(k_1, k_2)$ to ensure that $\omega$ is a stable attractor of dynamics (1) is a band between two parallel lines $k_1 + c(\omega)k_2 > d_1(\omega)$ and $k_1 + c(\omega)k_2 < d_2(\omega)$ (See regimes E and F in Fig.1(b)). It is noticed that as $d_1(\omega) > d_2(\omega)$, there is no regime of $(k_1, k_2)$ for the occurrence of that aRF $\omega$ as an attractor of equation (1). Therefore, the existence of such a structured aRF $\omega$ as an attractor of equation (1) is determined by $k_1$-intercept functions $d_1(\cdot)$ and $d_2(\cdot)$, and therefore by the covariance matrix $Q^{\mathcal{L}}$ or SDFs $r(\cdot)$ of all preceding layers.

## 3 Parameter Regimes For aRFs Between Layers $\mathcal{B}$ And $\mathcal{C}$

Based on our general theorems applicable to all layers, we mainly focus on describing the stabilization process of synaptic development from the 2nd ($\mathcal{B}$) to the 3rd layer ($\mathcal{C}$) by considering the effect of the system parameters on the weight development. For the sake of convenience, we assume that the input at 1st layer ($\mathcal{A}$) is independent normal distribution with mean 0 and variance 1, and the connection strengths from layer $\mathcal{A}$ to $\mathcal{B}$ are all-excitatory same as in Linsker's simulations. The emergence of various aRFs between layer $\mathcal{B}$ and $\mathcal{C}$ have been previously studied in the literature, and in this paper we mention only the following new results made possible by our approach:

(1) For the cell in layer $\mathcal{C}$, the all-excitatory and the all-inhibitory connection patterns still have the largest stable regimes. Denote both SDFs from layer $\mathcal{A}$ to $\mathcal{B}$ and from $\mathcal{B}$ to $\mathcal{C}$ as $r^{\mathcal{AB}}(\cdot, \cdot)$ and $r^{\mathcal{BC}}(\cdot)$ respectively. The parameter plane of $(k_1, k_2)$

Table 1: **The Principal Parameter Regimes**

| TYPE | PARAMETER REGIME | ATTRACTOR |
|---|---|---|
| Regime A | $k_1 + k_2 > d_1(+)$ (approx. $-k_1/k_2 < 1$) | All-excitatory aRF |
| Regime B | $k_1 - k_2 < d_2(-)$ (approx. $-k_1/k_2 > -1$) | All-inhibitory aRF |
| Regime C $= A \cap B$ | $k_1 + k_2 > d_1(+)$ and $k_1 - k_2 < d_2(-)$ | All-excitatory and all-inhibitory aRFs coexist |
| Regime D $= (A \cup B)^c$ | $k_2 < d_1(+) = -d_2(-)$ and approx. $-1 < -k_1/k_2 < 1$ | The structured aRFs may have separate parameter regimes |
| Regime E | $d_2(\omega) > k_1 + c(\omega)k_2 > d_1(\omega)$ where $c(\omega) > 0$ | Any connection pattern in which the excitatory connections constitute the majority |
| Regime F | $d_2(\omega) > k_1 + c(\omega)k_2 > d_1(\omega)$ where $c(\omega) < 0$ | Any connection pattern in which the inhibitory connections constitute the majority |
| Regime G $= E \cap F \cap A \cap B$ | $d_2(\omega^1) > k_1 + c(\omega^1)k_2 > d_1(\omega^1)$ ...... $d_2(\omega^\mu) > k_1 + c(\omega^\mu)k_2 > d_1(\omega^\mu)$ | A small coexistence regime of many connection patterns around the origin point of the parameter plane of $(k_1, k_2)$ |

is divided into four regimes by

$$k_1 + k_2 > d_1(+) = -\min_{1 \le i \le N_B} \sum_{j=1}^{N_B} \sum_{l=1}^{N_A} r^{AB}(i,l) r^{AB}(j,l) r^{BC}(j)$$

for all-excitatory pattern and

$$k_1 - k_2 < d_2(-) = \min_{1 \le i \le N_B} \sum_{j=1}^{N_B} \sum_{l=1}^{N_A} r^{AB}(i,l) r^{AB}(j,l) r^{BC}(j) = -d_1(+)$$

for all-inhibitory pattern (See Fig.1(a)).

(2) The parameter with large and negative $k_2$ and approximately $-1 < -k_1/k_2 < 1$ is favorable for the emergence of various structured connection patterns (e.g., ON-center cells, OFF-center cells, bi-lobed cells, and oriented cells). This is because this regime (See regime D in Fig.1) is removed from the parameter regime where both all-excitatory and all-inhibitory aRFs are dominant, including the coexistence regime of many kind of attractors around the origin point (See regime G in Fig.1(b)). The above results provide a precise picture about the principal parameter regimes summarized in Table 1.

(3) The relative size of the radiuses of two SDFs $r^{AB}(\cdot, \cdot)$ and $r^{BC}(\cdot)$ plays a key role in the evolution of various structured aRFs from $B$ to $C$. A given SDF $r^{LM}(i,j), i \in M, j \in L$ will be said to have a *range* $r_M$ if $r^{LM}(i,j)$ is 'sufficient small' for $||i-j|| \ge r_M$. For a Gaussian SDF $r^{LM}(j,k) \sim \exp(-||j-k||/r_M^2)$, $j \in L, k \in M$, the range $r_M$ is its standard deviation. We give the analytic prediction about the influence of the SDF's ranges $r_B, r_C$ on the dynamics by changing $r_B$ from the smallest extreme to the largest one with respect to $r_C$. For the smallest extreme of $r_B$ (i.e. the

synaptic connections from $\mathcal{A}$ to $\mathcal{B}$ are concentrated enough, and those from layer $\mathcal{B}$ to $\mathcal{C}$ are fully feedforward connected), we proved that any kind of connection pattern has a stable parameter regime and emerge under certain parameters, because each synaptic connection within an aRF is developed independently. As $r_B$ is changed from the smallest to the largest extreme, the development of synaptic connections between layer $\mathcal{B}$ and $\mathcal{C}$ will depend on each other stronger and stronger in the sense that most of connections have the same sign as their neighbors in an aRF. So for the largest extreme of $r_B$ (i.e. the weights from layer $\mathcal{A}$ to $\mathcal{B}$ are fully feedforward but there is no constraint on the SDF $r^{BC}(\cdot)$), any structured aRFs except for the all-excitatory and the all-inhibitory connection patterns will never arise at all, although there exist correlation in input activities (for a proof see Feng, Pan, & Roychowdhury, 1995). Therefore, without localized SDF, there would be no structured covariance matrix $\bar{Q} = \{[Q_{ij} + k_2]r(j)\}$ which embodies localized correlation in afferent activities. And without structured covariance matrix $\bar{Q}$, no structured aRFs would emerge.

(4)   As another application of our analyses, we present several numerical results on the parameter regimes of $(k_1, k_2, r_B, r_C)$ for the formation of various structured aRFs (Feng & Pan, 1993; Feng, Pan, & Roychowdhury, 1995) (where we assume that $r^{AB}(i,j) \sim \exp(-\|i-j\|/r_B^2)$, $i \in \mathcal{B}, j \in \mathcal{A}$, and $r^{BC}(i) \sim \exp(-\|i\|/r_C^2)$, $i \in \mathcal{B}$ as in (Linsker, 1986 & 1988)). For example, we show that various aRFs as attractors have different relative stability. For a fixed $r_C$, the SDF's range $r_B$ of the preceding layer as the third system parameter has various critical values for different attractors. That is, an attractor will no longer be stable if $r_B$ exceeds its corresponding critical value (See Fig. 2). For circularly symmetric ON-center cells, those aRFs with large ON-center core (which have positive or small negative slope value $c(\omega) \approx -k_1/k_2$) always have a stable parameter regime. But for those ON-center cells with large negative slope value $c(\omega)$, their stable parameter regimes decrease in size with $c(\omega)$. Similarly, circularly symmetric OFF-center cells with large OFF-center core (which have negative or small positive slope value $c(\omega)$) will be more stable than those with large positive average of weights. But for non-circularly-symmetric patterns (e.g., bi-lobed cells and oriented cells), only those attractors with zero average synaptic strength might always have a stable parameter regime (See regime H in Fig.1(b)). If the third parameter $r_B$ is large enough to exceed its critical values for other aRFs and $k_2$ is large and negative, then ON-center aRFs with positive $c(\omega)$ and OFF-center aRFs with negative $c(\omega)$ will be almost only attractors in regime D∩E and regime D∩F respectively. This conclusion makes it clear why we usually obtain ON-center aRFs in regime D∩E and OFF-center aRFs in regime D∩F much more easily than other patterns.

## 4   Concluding Remarks

One advantage of our rigorous approach to this kind of unsupervised Hebbian learning network is that, without approximation, it unifies the treatment of many diverse problems about dynamical mechanisms. It is important to notice that there is no assumption on the second item $h_i(\omega_\tau)$ on the right hand side of equation (1), and there is no restriction on the matrix $Q$. Our Theorems 1 and 2 provide the general framework for the description of the fixed point attractors for any difference equation of the type stated in (2) that uses a limiter function. Depending on the

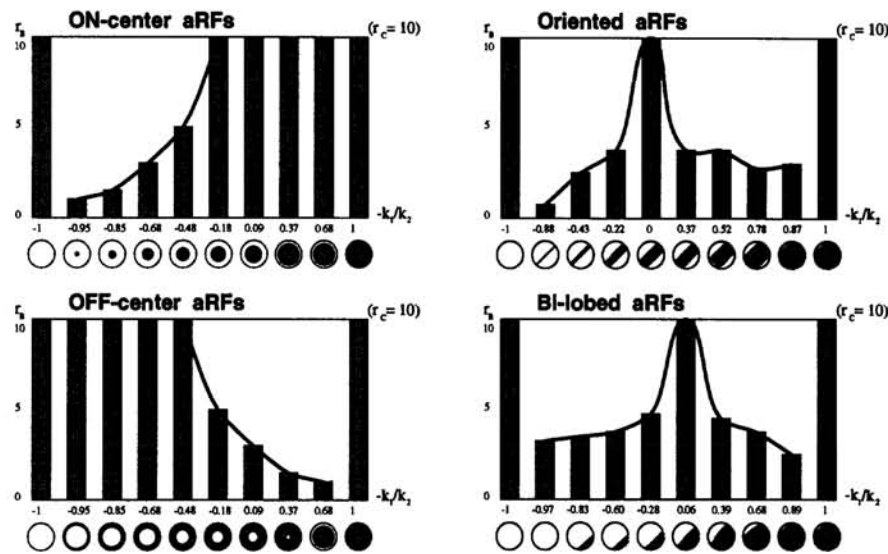

Figure 2: The critical values of the SDF's range $r_B$ for different connection patterns.

structure of the second item, $h_i(\omega_\tau)$, it is not difficult to adapt our Theorem 3 to obtain the precise relationship among system parameters in other kind of models as long as $f(\cdot)$ is a limiter function. Since the functions in the necessary and sufficient condition are computable (like our slope and $k_1$-intercept functions), one is always able to check whether a designated fixed point is stable for a specific set of parameters.

## Acknowledgements

The work of V. P. Roychowdhury and H. Pan was supported in part by the General Motors Faculty Fellowship and by the NSF Grant No. ECS-9308814. J. Feng was partially supported by Chinese National Key Project of Fundamental Research "Climbing Program" and CNR of Italy.

## Footnotes

[1]The SDF is explicitly incorporated into the dynamics (1) which is equivalent to

## References

R. Linsker. (1986) From basic network principle to neural architecture (series). *Proc. Natl. Acad. Sci. USA* **83**: 7508-7512, 8390-8394, 8779-8783.

R. Linsker. (1988) Self-organization in a perceptual network. *Computer* **21**(3): 105-117.

D. MacKay, & K. Miller. (1990) Analysis of Linsker's application of Hebbian rules to linear networks. *Network* **1**: 257-297.

J. Feng, & H. Pan. (1993) Analysis of Linsker-type Hebbian learning: Rigorous results. *Proc. 1993 IEEE Int. Conf. on Neural Networks - San Francisco Vol. III*, 1516-1521. Piscataway, NJ: IEEE.

J. Feng, H. Pan, & V. P. Roychowdhury. (1995) Linsker-type Hebbian learning: A qualitative analysis on the parameter space. (submitted).